# Cyclic Equilibria in Markov Games

**Martin Zinkevich and Amy Greenwald**
Department of Computer Science
Brown University
Providence, RI  02912
{maz,amy}@cs.brown.edu

**Michael L. Littman**
Department of Computer Science
Rutgers, The State University of NJ
Piscataway, NJ  08854–8019
mlittman@cs.rutgers.edu

## Abstract

Although variants of value iteration have been proposed for finding Nash or correlated equilibria in general-sum Markov games, these variants have not been shown to be effective in general. In this paper, we demonstrate by construction that existing variants of value iteration cannot find stationary equilibrium policies in arbitrary general-sum Markov games. Instead, we propose an alternative interpretation of the output of value iteration based on a new (non-stationary) equilibrium concept that we call "cyclic equilibria." We prove that value iteration identifies cyclic equilibria in a class of games in which it fails to find stationary equilibria. We also demonstrate empirically that value iteration finds cyclic equilibria in nearly all examples drawn from a random distribution of Markov games.

## 1   Introduction

Value iteration (Bellman, 1957) has proven its worth in a variety of sequential-decision-making settings, most significantly single-agent environments (Puterman, 1994), team games, and two-player zero-sum games (Shapley, 1953). In value iteration for Markov decision processes and team Markov games, the value of a state is defined to be the maximum over all actions of the value of the combination of the state and action (or *Q value*). In zero-sum environments, the max operator becomes a minimax over joint actions of the two players. Learning algorithms based on this update have been shown to compute equilibria in both model-based scenarios (Brafman & Tennenholtz, 2002) and Q-learning-like model-free scenarios (Littman & Szepesvári, 1996).

The theoretical and empirical success of such algorithms has led researchers to apply the same approach in general-sum games, in spite of exceedingly weak guarantees of convergence (Hu & Wellman, 1998; Greenwald & Hall, 2003). Here, value-update rules based on select Nash or correlated equilibria have been evaluated empirically and have been shown to perform reasonably in some settings. None has been identified that computes equilibria in general, however, leaving open the question of whether such an update rule is even possible.

Our main negative theoretical result is that an entire class of value-iteration update rules, including all those mentioned above, can be excluded from consideration for computing stationary equilibria in general-sum Markov games. Briefly, existing value-iteration algorithms compute Q values as an intermediate result, then derive policies from these Q

values. We demonstrate a class of games in which Q values, even those corresponding to an equilibrium policy, contain insufficient information for reconstructing an equilibrium policy.

Faced with the impossibility of developing algorithms along the lines of traditional value iteration that find stationary equilibria, we suggest an alternative equilibrium concept—cyclic equilibria. A cyclic equilibrium is a kind of non-stationary joint policy that satisfies the standard conditions for equilibria (no incentive to deviate unilaterally). However, unlike conditional non-stationary policies such as tit-for-tat and finite-state strategies based on the "folk theorem" (Osborne & Rubinstein, 1994), cyclic equilibria cycle rigidly through a set of stationary policies.

We present two positive results concerning cyclic equilibria. First, we consider the class of two-player two-state two-action games used to show that Q values cannot reconstruct all stationary equilibrium. Section 4.1 shows that value iteration finds cyclic equilibria for all games in this class. Second, Section 5 describes empirical results on a more general set of games. We find that on a significant fraction of these games, value iteration updates fail to converge. In contrast, value iteration finds cyclic equilibria for nearly all the games.

The success of value iteration in finding cyclic equilibria suggests this generalized solution concept could be useful for constructing robust multiagent learning algorithms.

## 2 An Impossibility Result for Q Values

In this section, we consider a subclass of Markov games in which transitions are deterministic and are controlled by one player at a time. We show that this class includes games that have no deterministic equilibrium policies. For this class of games, we present (proofs available in an extended technical report) two theorems. The first, a negative result, states that the Q values used in existing value-iteration algorithms are insufficient for deriving equilibrium policies. The second, presented in Section 4.1, is a positive result that states that value iteration does converge to cyclic equilibrium policies in this class of games.

### 2.1 Preliminary Definitions

Given a finite set $X$, define $\Delta(X)$ to be the set of all probability distributions over $X$.

**Definition 1** *A **Markov game** $\Gamma = [S, N, \mathbf{A}, T, R, \gamma]$ is a set of states $S$, a set of players $N = \{1, \ldots, n\}$, a set of actions for each player in each state $\{A_{i,s}\}_{s \in S, i \in N}$ (where we represent the set of all state-action pairs as $\mathbf{A} \equiv \bigcup_{s \in S} \left(\{s\} \times \prod_{i \in N} A_{i,s}\right)$), a transition function $T : \mathbf{A} \to \Delta(S)$, a reward function $R : \mathbf{A} \to \mathbf{R}^n$, and a discount factor $\gamma$.*

Given a Markov game $\Gamma$, let $A_s = \prod_{i \in N} A_{i,s}$. A **stationary policy** is a set of distributions $\{\pi(s) : s \in S\}$, where for all $s \in S$, $\pi(s) \in \Delta(A_s)$. Given a stationary policy $\pi$, define $V^{\pi,\Gamma} : S \to \mathbf{R}^n$ and $Q^{\pi,\Gamma} : \mathbf{A} \to \mathbf{R}^n$ to be the unique pair of functions satisfying the following system of equations: for all $i \in N$, for all $(s, a) \in \mathbf{A}$,

$$V_i^{\pi,\Gamma}(s) = \sum_{a \in A_s} \pi(s)(a) Q_i^{\pi,\Gamma}(s, a), \tag{1}$$

$$Q_i^{\pi,\Gamma}(s, a) = R_i(s, a) + \gamma \sum_{s' \in S} T(s, a)(s') V_i^{\pi,\Gamma}(s'). \tag{2}$$

A **deterministic** Markov game is a Markov game $\Gamma$ where the transition function is deterministic: $T : \mathbf{A} \to S$. A **turn-taking game** is a Markov game $\Gamma$ where in every state, only one player has a choice of action. Formally, for all $s \in S$, there exists a player $i \in N$ such that for all other players $j \in N \backslash \{i\}$, $|A_{j,s}| = 1$.

## 2.2 A Negative Result for Stationary Equilibria

A **NoSDE** (pronounced "nasty") game is a deterministic turn-taking Markov game $\Gamma$ with two players, two states, no more than two actions for either player in either state, and no deterministic stationary equilibrium policy. That the set of NoSDE games is non-empty is demonstrated by the game depicted in Figure 1. This game has no deterministic stationary equilibrium policy: If Player 1 sends, Player 2 prefers to send; but, if Player 2 sends, Player 1 prefers to keep; and, if Player 1 keeps, Player 2 prefers to keep; but, if Player 2 keeps, Player 1 prefers to send. No deterministic policy is an equilibrium because one player will always have an incentive to change policies.

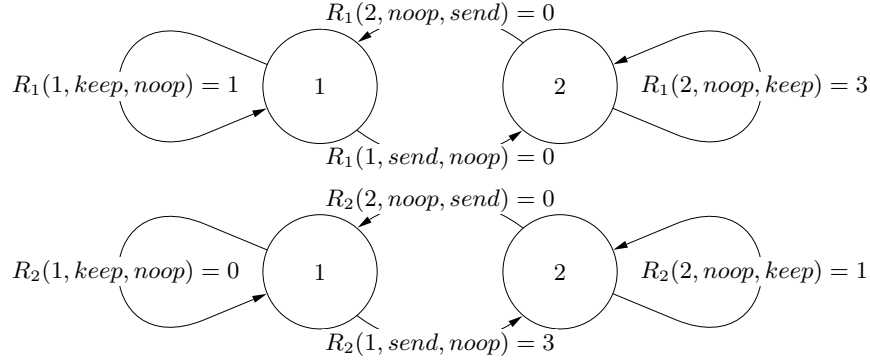

Figure 1: An example of a NoSDE game. Here, $S = \{1,2\}$, $A_{1,1} = A_{2,2} = \{keep, send\}$, $A_{1,2} = A_{2,1} = \{noop\}$, $T(1, keep, noop) = 1$, $T(1, send, noop) = 2$, $T(2, noop, keep) = 2$, $T(2, noop, send) = 1$, and $\gamma = 3/4$. In the unique stationary equilibrium, Player 1 sends with probability $2/3$ and Player 2 sends with probability $5/12$.

**Lemma 1** *Every NoSDE game has a* unique *stationary equilibrium policy.*[1]

It is well known that, in general Markov games, random policies are sometimes needed to achieve an equilibrium. This fact can be demonstrated simply by a game with one state where the utilities correspond to a bimatrix game with no deterministic equilibria (penny matching, say). Random actions in these games are sometimes linked with strategies that use "faking" or "bluffing" to avoid being exploited. That NoSDE games exist is surprising, in that randomness is needed even though actions are always taken with complete information about the other player's choice and the state of the game. However, the next result is even more startling. Current value-iteration algorithms attempt to find the Q values of a game with the goal of using these values to find a stationary equilibrium of the game. The main theorem of this paper states that it is not possible to derive a policy from the Q values for NoSDE games, and therefore in general Markov games.

**Theorem 1** *For any NoSDE game $\Gamma = [S, N, \mathbf{A}, T, R]$ with a unique equilibrium policy $\pi$, there exists another NoSDE game $\Gamma' = [S, N, \mathbf{A}, T, R']$ with its own unique equilibrium policy $\pi'$ such that $Q^{\pi,\Gamma} = Q^{\pi',\Gamma'}$ but $\pi \neq \pi'$ and $V^{\pi,\Gamma} \neq V^{\pi',\Gamma'}$.*

This result establishes that computing or learning Q values is insufficient to compute a stationary equilibrium of a game.[2] In this paper we suggest an alternative, where we still

do value iteration in the same way, but we extract a *cyclic equilibrium* from the sequence of values instead of a stationary one.

## 3 A New Goal: Cyclic Equilibria

A **cyclic policy** is a finite sequence of stationary policies $\pi = \{\pi_1, \ldots, \pi_k\}$. Associated with $\pi$ is a sequence of value functions $\{V^{\pi,\Gamma,j}\}$ and Q-value functions $\{Q^{\pi,\Gamma,j}\}$ such that

$$V_i^{\pi,\Gamma,j}(s) = \sum_{a \in A_s} \pi_j(s)(a) \, Q_i^{\pi,\Gamma,j}(s,a) \quad \text{and} \tag{3}$$

$$Q_i^{\pi,\Gamma,j}(s,a) = R_i(s,a) + \gamma \sum_{s' \in S} T(s,a)(s') \, V_i^{\pi,\Gamma,inc_k(j)}(s') \tag{4}$$

where for all $j \in \{1, \ldots, k\}$, $inc_k(k) = 1$ and $inc_k(j) = j+1$ if $j < k$.

**Definition 2** *Given a Markov game $\Gamma$, a **cyclic correlated equilibrium** is a cyclic policy $\pi$, where for all $j \in \{1, \ldots, k\}$, for all $i \in N$, for all $s \in S$, for all $a_i, a_i' \in A_{i,s}$:*

$$\sum_{a_{-i} \in A_{-i,s}} \pi_j(s)(a_i, a_{-i}) \, Q_i^{\pi,\Gamma,j}(s, a_i, a_{-i}) \geq \sum_{(a_i, a_{-i}) \in A_s} \pi_j(s)(a_i, a_{-i}) \, Q^{\pi,\Gamma,j}(s, a_i', a_{-i}). \tag{5}$$

Here, $a_{-i}$ denotes a joint action for all players except $i$. A similar definition can be constructed for Nash equilibria by insisting that all policies $\pi_j(s)$ are product distributions. In Definition 2, we imagine that action choices are moderated by a referee with a clock that indicates the current stage $j$ of the cycle. At each stage, a typical correlated equilibrium is executed, meaning that the referee chooses a joint action $a$ from $\pi_j(s)$, tells each agent its part of that joint action, and no agent can improve its value by eschewing the referee's advice. If no agent can improve its value by more than $\epsilon$ at any stage, we say $\pi$ is an **$\epsilon$-correlated cyclic equilibrium**.

A **stationary correlated equilibrium** is a cyclic correlated equilibrium with $k = 1$. In the next section, we show how value iteration can be used to derive cyclic correlated equilibria.

## 4 Value Iteration in General-Sum Markov Games

For a game $\Gamma$, define $\mathcal{Q}_\Gamma = (\mathbf{R}^n)^{\mathbf{A}}$ to be the set of all state-action (**Q**) value functions, $\mathcal{V}_\Gamma = (\mathbf{R}^n)^S$ to be the set of all value functions, and $\Pi_\Gamma$ to be the set of all stationary policies. Traditionally, value iteration can be broken down into estimating a Q value based upon a value function, selecting a policy $\pi$ given the Q values, and deriving a value function based upon $\pi$ and the Q value functions. Whereas the first and the last step are fairly straightforward, the step in the middle is quite tricky. A pair $(\pi, Q) \in \Pi_\Gamma \times \mathcal{Q}_\Gamma$ **agree** (see Equation 5) if, for all $s \in S, i \in N, a_i, a_i' \in A_{i,s}$:

$$\sum_{a_{-i} \in A_{-i,s}} \pi(s)(a_i, a_{-i}) \, Q_i(s, a_i, a_{-i}) \geq \sum_{(a_i, a_{-i}) \in A_s} \pi(s)(a_i, a_{-i}) \, Q(s, a_i', a_{-i}). \tag{6}$$

Essentially, $Q$ and $\pi$ agree if $\pi$ is a best response for each player given payoffs $Q$. An **equilibrium-selection rule** is a function $f : \mathcal{Q}_\Gamma \rightarrow \Pi_\Gamma$ such that for all $Q \in \mathcal{Q}_\Gamma$, $(f(Q), Q)$ agree. The set of all such rules is $F_\Gamma$. In essence, these rules update values assuming an equilibrium policy for a one-stage game with $Q(s,a)$ providing the terminal rewards. Examples of equilibrium-selection rules are best-Nash, utilitarian-CE, dictatorial-CE, plutocratic-CE, and egalitarian-CE (Greenwald & Hall, 2003). (Utilitarian-CE, which we return to later, selects the correlated equilibrium in which total of the payoffs is maximized.) Foe-VI and Friend-VI (Littman, 2001) do not fit into our formalism, but it can

be proven that in NoSDE games they converge to deterministic policies that are neither stationary nor cyclic equilibria. Define $d_\Gamma : \mathcal{V}_\Gamma \times \mathcal{V}_\Gamma \to \mathbf{R}$ to be a distance metric over value functions, such that

$$d_\Gamma(V, V') = \max_{s \in S, i \in N} |V_i(s) - V'_i(s)|. \tag{7}$$

Using our notation, the value-iteration algorithm for general-sum Markov games can be described as follows.

---

**Algorithm 1: ValueIteration**(game $\Gamma$, $V^0 \in \mathcal{V}_\Gamma$, $f \in \mathbf{F}_\Gamma$, Integer $T$)

For $t := 1$ to $T$:

    1. $\forall s \in S, a \in A, Q^t(s,a) := R(s,a) + \gamma \sum_{s' \in S} T(s,a)(s') V^{t-1}(s')$.

    2. $\pi^t = f(Q^t)$.

    3. $\forall s \in S, V^t(s) = \sum_{a \in A_s} \pi^t(s)(a) Q^t(s,a)$.

Return $\{Q^1, \ldots, Q^T\}, \{\pi^1, \ldots, \pi^T\}, \{V^1, \ldots, V^T\}$.

---

If a stationary equilibrium is sought, the final policy is returned.

---

**Algorithm 2: GetStrategy**(game $\Gamma$, $V^0 \in \mathcal{V}_\Gamma$, $f \in \mathbf{F}_\Gamma$, Integer $T$)

    1. Run $(Q^1 \ldots Q^T, \pi^1 \ldots \pi^T, V^1 \ldots V^T) = \text{ValueIteration}(\Gamma, V^0, f, T)$.

    2. Return $\pi^T$.

---

For cyclic equilibria, we have a variety of options for how many past stationary policies we want to consider for forming a cycle. Our approach searches for a recent value function that matches the final value function (an exact match would imply a true cycle). Ties are broken in favor of the shortest cycle length. Observe that the order of the policies returned by value iteration is reversed to form a cyclic equilibrium.

---

**Algorithm 3: GetCycle**(game $\Gamma$, $V^0 \in \mathcal{V}_\Gamma$, $f \in \mathbf{F}_\Gamma$, Integer $T$, Integer $maxCycle$)

    1. If $maxCycle \geq T$, $maxCycle := T - 1$.

    2. Run $(Q^1 \ldots Q^T, \pi^1 \ldots \pi^T, V^1 \ldots V^T) = \text{ValueIteration}(\Gamma, V^0, f, T)$.

    3. Define $k := \text{argmin}_{t \in \{1, \ldots, maxCycle\}} d(V^T, V^{T-t})$.

    4. For each $t \in \{1, \ldots, k\}$ set $\pi_t := \pi^{T+1-t}$.

---

## 4.1 Convergence Conditions

**Fact 1** *If $d(V^T, V^{T-1}) = \epsilon$ in **GetStrategy**, then **GetStrategy** returns an $\frac{\epsilon\gamma}{1-\gamma}$-correlated equilibrium.*

**Fact 2** *If **GetCycle** returns a cyclic policy of length $k$ and $d(V^T, V^{T-k}) = \epsilon$, then **GetCycle** returns an $\frac{\epsilon\gamma}{1-\gamma^k}$-correlated cyclic equilibrium.*

Since, given $V^0$ and $\Gamma$, the space of value functions is bounded, *eventually* there will be two value functions in $\{V^1, \ldots, V^T\}$ that are close according to $d_\Gamma$. Therefore, the two practical (and open) questions are (1) how many iterations does it take to find an $\epsilon$-correlated cyclic equilibrium? and (2) How large is the cyclic equilibrium that is found?

In addition to approximate convergence described above, in two-player turn-taking games, one can prove *exact convergence*. In fact, all the members of $\mathbf{F}_\Gamma$ described above can be construed as generalizations of utilitarian-CE in turn-taking games, and utilitarian-CE is proven to converge.

**Theorem 2** *Given the utilitarian-CE equilibrium-selection rule $f$, for every NoSDE game $\Gamma$, for every $V^0 \in \mathcal{V}_\Gamma$, there exists some finite $T$ such that* **GetCycle**$(\Gamma, V^0, f, T, \lceil T/2 \rceil)$ *returns a cyclic correlated equilibrium.*

Theoretically, we can imagine passing infinity as a parameter to value iteration. Doing so shows the limitation of value-iteration in Markov games.

**Theorem 3** *Given the utilitarian-CE equilibrium-selection rule $f$, for any NoSDE game $\Gamma$ with unique equilibrium $\pi$, for every $V^0 \in \mathcal{V}_\Gamma$, the value-function sequence $\{V^1, V^2, \ldots\}$ returned from* **ValueIteration**$(\Gamma, V^0, f, \infty)$ *does not converge to $V^\pi$.*

Since all of the other rules specified above (except friend-VI and foe-VI) can be implemented with the utilitarian-CE equilibrium-selection rule, none of these rules will be guaranteed to converge, even in such a simple class as turn-taking games!

**Theorem 4** *Given the game $\Gamma$ in Figure 1 and its stationary equilibrium $\pi$, given $V_i^0(s) = 0$ for all $i \in N$, $s \in S$, then for any update rule $f \in \mathbf{F}_\Gamma$, the value-function sequence $\{V^1, V^2, \ldots\}$ returned from* **ValueIteration**$(\Gamma, V^0, f, \infty)$ *does not converge to $V^\pi$.*

## 5 Empirical Results

To complement the formal results of the previous sections, we ran two batteries of tests on value iteration in randomly generated games. We assessed the convergence behavior of value iteration to stationary and cyclic equilibria.

### 5.1 Experimental Details

Our game generator took as input the set of players $N$, the set of states $S$, and for each player $i$ and state $s$, the actions $A_{i,s}$. To construct a game, for each state-joint action pair $(s, a) \in \mathbf{A}$, for each agent $i \in N$, the generator sets $R_i(s, a)$ to be an integer between 0 and 99, chosen uniformly at random. Then, it selects $T(s, a)$ to be deterministic, with the resulting state chosen uniformly at random. We used a consistent discount factor of $\gamma = 0.75$ to decrease experimental variance.

The primary dependent variable in our results was the frequency with which value iteration converged to a stationary Nash equilibrium or a cyclic Nash equilibrium (of length less than 100). To determine convergence, we first ran value iteration for 1000 steps. If $d_\Gamma(V^{1000}, V^{999}) \leq 0.0001$, then we considered value iteration to have converged to a stationary policy. If for some $k \leq 100$

$$\max_{t \in \{1, \ldots, k\}} d_\Gamma(V^{1001-t}, V^{1001-(t+k)}) \leq 0.0001, \tag{8}$$

then we considered value iteration to have converged to a cycle.[3]

To determine if a game has a deterministic equilibrium, for every deterministic policy $\pi$, we ran policy evaluation (for 1000 iterations) to estimate $V^{\pi,\Gamma}$ and $Q^{\pi,\Gamma}$, and then checked if $\pi$ was an $\epsilon$-correlated equilibrium for $\epsilon = 0.0001$.

### 5.2 Turn-taking Games

In the first battery of tests, we considered sets of turn-taking games with $x$ states and $y$ actions: formally, there were $x$ states $\{1, \ldots, x\}$. In odd-numbered states, Player 1 had $y$

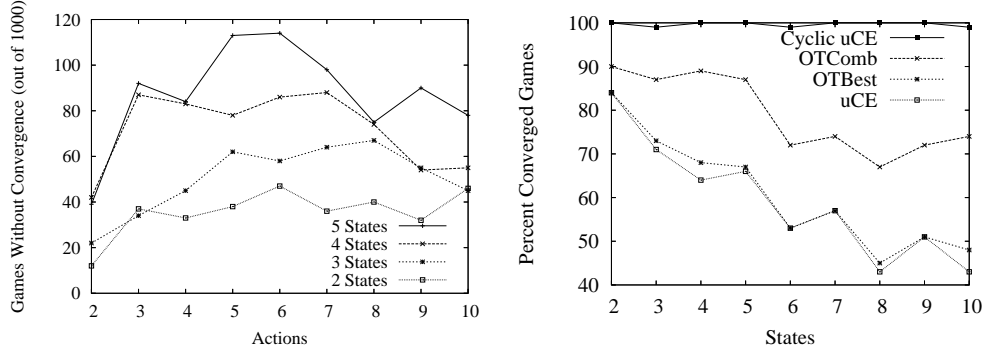

Figure 2: (Left) For each combination of states and actions, 1000 deterministic turn-taking games were generated. The graph plots the number of games where value iteration did not converge to a stationary equilibrium. (Right) Frequency of convergence on 100 randomly generated games with simultaneous actions. **Cyclic uCE** is the number of times utilitarian-CE converged to a cyclic equilibrium. **OTComb** is the number of games where any one of Friend-VI, Foe-VI, utilitarian-NE-VI, and 5 variants of correlated equilibrium-VI: dictatorial-CE-VI (First Player), dictatorial-CE-VI (Second Player), utilitarian-CE-VI, plutocratic-CE-VI, and egalitarian-VI converged to a stationary equilibrium. **OTBest** is the maximum number of games where the best fixed choice of the equilibrium-selection rule converged. **uCE** is the number of games in which utilitarian-CE-VI converged to a stationary equilibrium.

actions and Player 2 had one action: in even-numbered states, Player 1 had one action and Player 2 had $y$ actions. We varied $x$ from 2 to 5 and $y$ from 2 to 10. For each setting of $x$ and $y$, we generated and tested one thousand games.

Figure 2 (left) shows the number of generated games for which value iteration did *not* converge to a stationary equilibrium. We found that nearly half (48%, as many as 5% of the total set) of these non-converged games had no stationary, deterministic equilibria (they were NoSDE games). The remainder of the stationary, deterministic equilibria were simply not discovered by value iteration. We also found that value iteration converged to cycles of length 100 or less in 99.99% of the games.

### 5.3 Simultaneous Games

In a second set of experiments, we generated two-player Markov games where both agents have at least two actions in every state. We varied the number of states between 2 and 9, and had either 2 or 3 actions for every agent in every state.

Figure 2 (right) summarizes results for 3-action games (2-actions games were qualitatively similar, but converged more often). Note that the fraction of random games on which the algorithms converged to stationary equilibria decreases as the number of states increases. This result holds because the larger the game, the larger the chance that value iteration will fall into a cycle on some subset of the states. Once again, we see that the cyclic equilibria are found much more reliably than stationary equilibria by value-iteration algorithms. For example, utilitarian-CE converges to a cyclic correlated equilibrium about 99% of the time, whereas with 10 states and 3 actions, on 26% of the games none of the techniques converge.

# 6 Conclusion

In this paper, we showed that value iteration, the algorithmic core of many multiagent planning reinforcement-learning algorithms, is not well behaved in Markov games. Among other impossibility results, we demonstrated that the Q-value function retains too little information for constructing optimal policies, even in 2-state, 2-action, deterministic turn-taking Markov games. In fact, there are an infinite number of such games with different Nash equilibrium value functions that have identical Q-value functions. This result holds for proposed variants of value iteration from the literature such as updating via a correlated equilibrium or a Nash equilibrium, since, in turn-taking Markov games, both rules reduce to updating via the action with the maximum value for the controlling player.

Our results paint a bleak picture for the use of value-iteration-based algorithms for computing stationary equilibria. However, in a class of games we called NoSDE games, a natural extension of value iteration converges to a limit cycle, which is in fact a cyclic (nonstationary) Nash equilibrium policy. Such cyclic equilibria can also be found reliably for randomly generated games and there is evidence that they appear in some naturally occurring problems (Tesauro & Kephart, 1999). One take-away message of our work is that nonstationary policies may hold the key to improving the robustness of computational approaches to planning and learning in general-sum games.

### Acknowledgements

This research was supported by NSF Grant #IIS-0325281, NSF Career Grant #IIS-0133689, and the Alberta Ingenuity Foundation through the Alberta Ingenuity Centre for Machine Learning.

## Footnotes

[1] The policy is both a correlated equilibrium and a Nash equilibrium.

[2] Although maintaining Q values *and* state values and deriving policies from both sets of functions might circumvent this problem, we are not aware of existing value-iteration algorithms or learning algorithms that do so. This observation presents a possible avenue of research not followed in this paper.

[3]In contrast to the **GetCycle** algorithm, we are here concerned with finding a cyclic equilibrium so we check an entire cycle for convergence.

# References

Bellman, R. (1957). *Dynamic programming*. Princeton, NJ: Princeton University Press.

Brafman, R. I., & Tennenholtz, M. (2002). R-MAX—a general polynomial time algorithm for near-optimal reinforcement learning. *Journal of Machine Learning Research*, *3*, 213–231.

Greenwald, A., & Hall, K. (2003). Correlated *Q*-learning. *Proceedings of the Twentieth International Conference on Machine Learning* (pp. 242–249).

Hu, J., & Wellman, M. (1998). Multiagent reinforcement learning:theoretical framework and an algorithm. *Proceedings of the Fifteenth International Conference on Machine Learning* (pp. 242–250). Morgan Kaufman.

Littman, M. (2001). Friend-or-foe Q-learning in general-sum games. *Proceedings of the Eighteenth International Conference on Machine Learning* (pp. 322–328). Morgan Kaufmann.

Littman, M. L., & Szepesvári, C. (1996). A generalized reinforcement-learning model: Convergence and applications. *Proceedings of the Thirteenth International Conference on Machine Learning* (pp. 310–318).

Osborne, M. J., & Rubinstein, A. (1994). *A Course in Game Theory*. The MIT Press.

Puterman, M. (1994). *Markov decision processes: Discrete stochastic dynamic programming*. Wiley-Interscience.

Shapley, L. (1953). Stochastic games. *Proceedings of the National Academy of Sciences of the United States of America*, *39*, 1095–1100.

Tesauro, G., & Kephart, J. (1999). Pricing in agent economies using multi-agent Q-learning. *Proceedings of Fifth European Conference on Symbolic and Quantitative Approaches to Reasoning with Uncertainty* (pp. 71–86).
